# Spectral Relaxation for K-means Clustering

**Hongyuan Zha & Xiaofeng He**
Dept. of Comp. Sci. & Eng.
The Pennsylvania State University
University Park, PA 16802
*{zha,xhe}@cse.psu.edu*

**Chris Ding & Horst Simon**
NERSC Division
Lawrence Berkeley National Lab.
UC Berkeley, Berkeley, CA 94720
*{chqding,hdsimon}@lbl.gov*

**Ming Gu**
Dept. of Mathematics
UC Berkeley, Berkeley, CA 95472
*mgu@math.berkeley.edu*

## Abstract

The popular K-means clustering partitions a data set by minimizing a sum-of-squares cost function. A coordinate descend method is then used to find local minima. In this paper we show that the minimization can be reformulated as a trace maximization problem associated with the Gram matrix of the data vectors. Furthermore, we show that a relaxed version of the trace maximization problem possesses *global* optimal solutions which can be obtained by computing a partial eigendecomposition of the Gram matrix, and the cluster assignment for each data vectors can be found by computing a pivoted QR decomposition of the eigenvector matrix. As a by-product we also derive a *lower* bound for the minimum of the sum-of-squares cost function.

## 1   Introduction

K-means is a very popular method for general clustering [6]. In K-means clusters are represented by centers of mass of their members, and it can be shown that the K-means algorithm of alternating between assigning cluster membership for each data vector to the nearest cluster center and computing the center of each cluster as the centroid of its member data vectors is equivalent to finding the minimum of a sum-of-squares cost function using coordinate descend. Despite the popularity of K-means clustering, one of its major drawbacks is that the coordinate descend search method is prone to local minima. Much research has been done on computing refined initial points and adding explicit constraints to the sum-of-squares cost function for K-means clustering so that the search can converge to better local minimum [1, 2]. In this paper we tackle the problem from a different angle: we find an equivalent formulation of the sum-of-squares minimization as a trace maximization problem with special constraints; relaxing the constraints leads to a maximization problem

that possesses optimal *global* solutions. As a by-product we also have an easily computable *lower* bound for the minimum of the sum-of-squares cost function. Our work is inspired by [9, 3] where connection to Gram matrix and extension of K-means method to general Mercer kernels were investigated.

The rest of the paper is organized as follows: in section 2, we derive the equivalent trace maximization formulation and discuss its spectral relaxation. In section 3, we discuss how to assign cluster membership using pivoted QR decomposition, taking into account the special structure of the partial eigenvector matrix. Finally, in section 4, we illustrate the performance of the clustering algorithms using document clustering as an example.

**Notation.** Throughout, $\| \cdot \|$ denotes the Euclidean norm of a vector. The trace of a matrix $A$, i.e., the sum of its diagonal elements, is denoted as $\text{trace}(A)$. The Frobenius norm of a matrix $\|A\|_F = \sqrt{\text{trace}(A^T A)}$. $I_n$ denotes identity matrix of order $n$.

## 2   Spectral Relaxation

Given a set of $m$-dimensional data vectors $a_i, i = 1, \ldots, n$, we form the $m$-by-$n$ data matrix $A = [a_1, \ldots, a_n]$. A partition $\Pi$ of the date vectors can be written in the following form

$$AE = [A_1, \ldots, A_k], \quad A_i = [a_1^{(i)}, \ldots, a_{s_i}^{(i)}] \tag{1}$$

where $E$ is a permutation matrix, and $A_i$ is $m$-by-$s_i$, i.e., the $i$th cluster contains the data vectors in $A_i$. For a given partition $\Pi$ in (1), the associated sum-of-squares cost function is defined as

$$\text{ss}(\Pi) = \sum_{i=1}^{k} \sum_{s=1}^{s_i} \|a_s^{(i)} - m_i\|^2, \quad m_i = \sum_{s=1}^{s_i} a_s^{(i)} / s_i,$$

i.e., $m_i$ is the mean vector of the data vectors in cluster $i$. Let $e$ be a vector of appropriate dimension with all elements equal to one, it is easy to see that $m_i = A_i e / s_i$ and

$$\text{ss}_i \equiv \sum_{s=1}^{s_i} \|a_s^{(i)} - m_i\|^2 = \|A_i - m_i e^T\|_F^2 = \|A_i (I_{s_i} - ee^T / s_i)\|_F^2.$$

Notice that $I_{s_i} - ee^T / s_i$ is a projection matrix and $(I_{s_i} - ee^T / s_i)^2 = I_{s_i} - ee^T / s_i$, it follows that

$$\text{ss}_i = \text{trace}(A_i (I_{s_i} - ee^T / s_i) A_i^T) = \text{trace}((I_{s_i} - ee^T / s_i) A_i^T A_i).$$

Therefore,

$$\text{ss}(\Pi) = \sum_{i=1}^{k} \text{ss}_i = \sum_{i=1}^{k} \left( \text{trace}(A_i^T A_i) - \left( \frac{e^T}{\sqrt{s_i}} \right) A_i^T A_i \left( \frac{e}{\sqrt{s_i}} \right) \right).$$

Let the $n$-by-$k$ *orthonormal* matrix $X$ be

$$X = \begin{matrix} s_1 \\ s_2 \\ \vdots \\ s_k \end{matrix} \begin{pmatrix} e/\sqrt{s_1} & & & \\ & e/\sqrt{s_2} & & \\ & & \ddots & \\ & & & e/\sqrt{s_k} \end{pmatrix} \tag{2}$$

The sum-of-squares cost function can now be written as

$$\mathrm{ss}(\Pi) = \mathrm{trace}(A^T A) - \mathrm{trace}(X^T A^T A X),$$

and its minimization is equivalent to

$$\max\{\ \mathrm{trace}(X^T A^T A X)\ |\ X \text{ of the form in (2)}\}.$$

REMARK. Without loss of generality, let $E = I$ in (1). If we let $x_i$ be the cluster indicator vector, i.e.,

$$x_i^T = [0, \ldots, 0, \underbrace{1, \ldots, 1}_{s_i}, 0, \ldots, 0].$$

Then it is easy to see that

$$\mathrm{trace}(X^T A^T A X) = \sum_{i=1}^{k} \frac{x_i^T A^T A x_i}{x_i^T x_i} = \sum_{i=1}^{k} \frac{\|A x_i\|^2}{\|x_i\|^2}.$$

Using the partition in (1), the right-hand side of the above can be written as

$$\sum_{i=1}^{k} s_i \left\| \frac{A_i e}{s_i} \right\|^2 = \sum_{i=1}^{k} s_i \|m_i\|^2,$$

a weighted sum of the squared Euclidean norms of the mean vector of each clusters.

REMARK. If we consider the elements of the Gram matrix $A^T A$ as measuring similarity between data vectors, then we have shown that Euclidean distance leads to Euclidean inner-product similarity. This inner-product can be replaced by a general Mercer kernel as is done in [9, 3].

Ignoring the special structure of $X$ and let it be an arbitrary orthonormal matrix, we obtain a *relaxed* maximization problem

$$\max_{X^T X = I_k} \mathrm{trace}(X^T A^T A X) \qquad (3)$$

It turns out the above trace maximization problem has a closed-form solution.

**Theorem.** *(Ky Fan) Let H be a symmetric matrix with eigenvalues*

$$\lambda_1 \geq \lambda_2 \geq \cdots \geq \lambda_n,$$

*and the corresponding eigenvectors $U = [u_1, \ldots, u_n]$. Then*

$$\lambda_1 + \cdots \lambda_k = \max_{X^T X = I_k} \mathrm{trace}(X^T H X).$$

*Moreover, the optimal $X^*$ is given by $X^* = [u_1, \ldots, u_k]Q$ with $Q$ an arbitrary orthogonal matrix.*

It follows from the above theorem that we need to compute the largest $k$ eigenvectors of the Gram matrix $A^T A$. As a by-product, we have

$$\min_{\Pi} \mathrm{ss}(\Pi) \geq \mathrm{trace}(A^T A) - \max_{X^T X = I_k} \mathrm{trace}(X^T A^T A X) = \sum_{i=k+1}^{\min\{m,n\}} \sigma_i^2(A), \quad (4)$$

where $\sigma_i(A)$ is the $i$ largest singular value of $A$. This gives a *lower bound* for the minimum of the sum-of-squares cost function.

REMARK. It is easy to see from the above derivation that we can replace $A$ with $A - ae^T$, where $a$ is an arbitrary vector. Then we have the following lower bound

$$\min_{\Pi} \text{ss}(\Pi) \geq \max_{a} \sum_{i=k+1}^{\min\{m,n\}} \sigma_i^2(A - ae^T).$$

REMARK. One might also try the following approach: notice that

$$\|A_i - m_i e^T\|_F^2 = \frac{1}{2s_i} \sum_{a_j \in A_i} \sum_{a_{j'} \in A_i} \|a_j - a_{j'}\|^2.$$

Let $W = (\|a_i - a_j\|^2)_{i,j=1}^n$, and and $x_i = [x_{ij}]_{j=1}^n$ with

$$x_{ij} = \begin{cases} 1 & \text{if } a_j \in A_i \\ 0 & \text{otherwise} \end{cases}$$

Then

$$\text{ss}(\Pi) = \frac{1}{2} \sum_{i=1}^{k} \frac{x_i^T W x_i}{x_i^T x_i} \geq \frac{1}{2} \min_{Z^T Z = I_k} Z^T W Z = \frac{1}{2} \sum_{i=n-k+1}^{n} \lambda_i(W).$$

Unfortunately, some of the smallest eigenvalues of $W$ can be negative.

Let $X_k$ be the $n$-by-$k$ matrix consisting of the $k$ largest eigenvectors of $A^T A$. Each row of $X_k$ corresponds to a data vector, and the above process can be considered as transforming the original data vectors which live in a $m$-dimensional space to new data vectors which now live in a $k$-dimensional space. One might be attempted to compute the cluster assignment by applying the ordinary K-means method to those data vectors in the reduced dimension space. In the next section, we discuss an alternative that takes into account the structure of the eigenvector matrix $X_k$ [5].

REMARK. The similarity of the projection process to principal component analysis is deceiving: the goal here is not to reconstruct the data matrix using a low-rank approximation but rather to capture its cluster structure.

## 3  Cluster Assignment Using Pivoted QR Decomposition

Without loss of generality, let us assume that the best partition of the data vectors in $A$ that minimizes $\text{ss}(\Pi)$ is given by $A = [A_1, \ldots, A_k]$, each submatrix $A_i$ corresponding to a cluster. Now write the Gram matrix of $A$ as

$$A^T A = \begin{bmatrix} A_1^T A_1 & 0 & \cdots & 0 \\ 0 & A_2^T A_2 & \cdots & 0 \\ \vdots & \vdots & \cdots & \vdots \\ 0 & 0 & \cdots & A_k^T A_k \end{bmatrix} + E \equiv B + E.$$

If the overlaps among the clusters represented by the submatrices $A_i$ are small, then the norm of $E$ will be small as compare with the block diagonal matrix $B$ in the above equation. Let the largest eigenvector of $A_i^T A_i$ be $y_i$, and

$$A_i^T A_i y_i = \mu_i y_i, \quad \|y_i\| = 1, \quad i = 1, \ldots, k,$$

then the columns of the matrix

$$Y_k = \begin{matrix} s_1 \\ s_2 \\ \vdots \\ s_k \end{matrix} \begin{pmatrix} y_1 & & & \\ & y_2 & & \\ & & \ddots & \\ & & & y_k \end{pmatrix}$$

span an invariant subspace of $B$. Let the eigenvalues and eigenvectors of $A^T A$ be

$$\lambda_1 \geq \lambda_2 \geq \ldots \geq \lambda_n, \quad A^T A x_i = \lambda_i x_i, \quad i = 1, \ldots, n.$$

Assume that there is a gap between the two eigenvalue sets $\{\mu_1, \ldots \mu_k\}$ and $\{\lambda_{k+1}, \ldots \lambda_n\}$, i.e.,

$$0 < \delta = \min\{|\mu_i - \lambda_j| \mid i = 1, \ldots, k, j = k+1, \ldots, n\}.$$

Then Davis-Kahan $\sin(\Theta)$ theorem states that $\|Y_k^T [x_{k+1}, \ldots, x_n]\| \leq \|E\|/\delta$ [11, Theorem 3.4]. After some manipulation, it can be shown that

$$X_k \equiv [x_1, \ldots, x_k] = Y_k V + O(\|E\|),$$

where $V$ is an $k$-by-$k$ orthogonal matrix. Ignoring the $O(\|E\|)$ term, we see that

$$X_k^T = [\underbrace{y_{11} v_1, \ldots, y_{1s_1} v_1}_{\text{cluster } 1}, \ldots, \underbrace{y_{k1} v_k, \ldots, y_{ks_k} v_k}_{\text{cluster } k}],$$

where we have used $y_i^T = [y_{i1}, \ldots, y_{is_i}]$, and $V^T = [v_1, \ldots, v_k]$. A key observation is that all the $v_i$ are orthogonal to each other: once we have selected a $v_i$, we can jump to other clusters by looking at the orthogonal complement of $v_i$. Also notice that $\|y_i\| = 1$, so the elements of $y_i$ can not be all small. A robust implementation of the above idea can be obtained as follows: we pick a column of $X_k^T$ which has the largest norm, say, it belongs to cluster $i$, we orthogonalize the rest of the columns of $X_k^T$ against this column. For the columns belonging to cluster $i$ the residual vector will have small norm, and for the other columns the residual vectors will tend to be not small. We then pick another vector with the largest residual norm, and orthogonalize the other residual vectors against this residual vector. The process can be carried out $k$ steps, and it turns out to be exactly QR decomposition with column pivoting applied to $X_k^T$ [4], i.e., we find a permutation matrix $P$ such that

$$X_k^T P = QR = Q[R_{11}, R_{12}],$$

where $Q$ is a $k$-by-$k$ orthogonal matrix, and $R_{11}$ is a $k$-by-$k$ upper triangular matrix. We then compute the matrix

$$\hat{R} = R_{11}^{-1}[R_{11}, R_{12}]P^T = [I_k, R_{11}^{-1} R_{12}]P^T.$$

Then the cluster membership of each data vector is determined by the row index of the largest element *in absolute value* of the corresponding column of $\hat{R}$.

REMARK. Sometimes it may be advantageous to include more than $k$ eigenvectors to form $X_s^T$ with $s > k$. We can still use QR decomposition with column pivoting to select $k$ columns of $X_s^T$ to form an $s$-by-$k$ matrix, say $X$. Then for each column $z$ of $X_s^T$ we compute the least squares solution of $t^* = \operatorname{argmin}_{t \in \mathcal{R}^k} \|z - Xt\|$. Then the cluster membership of $z$ is determined by the row index of the largest element *in absolute value* of $t^*$.

## 4 Experimental Results

In this section we present our experimental results on clustering a dataset of newsgroup articles submitted to 20 newsgroups.[1] This dataset contains about 20,000 articles (email messages) evenly divided among the 20 newsgroups. We list the names of the newsgroups together with the associated group labels.

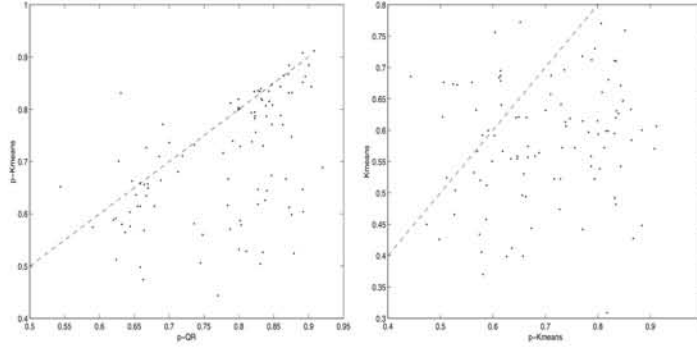

Figure 1: Clustering accuracy for five newsgroups NG2/NG9/NG10/NG15/NG18: p-QR vs. p-Kmeans (left) and p-Kmeans vs. Kmeans (right)

```
NG1: alt.atheism    NG2: comp.graphics
NG3: comp.os.ms-windows.misc    NG4: comp.sys.ibm.pc.hardware
NG5:comp.sys.mac.hardware    NG6: comp.windows.x
NG7:misc.forsale    NG8: rec.autos
NG9:rec.motorcycles    NG10: rec.sport.baseball
NG11:rec.sport.hockey    NG12: sci.crypt
NG13:sci.electronics    NG14: sci.med
NG15:sci.space    NG16: soc.religion.christian
NG17:talk.politics.guns    NG18: talk.politics.mideast
NG19:talk.politics.misc    NG20: talk.religion.misc
```

We used the *bow* toolkit to construct the term-document matrix for this dataset, specifically we use the tokenization option so that the UseNet headers are stripped, and we also applied stemming [8]. The following three preprocessing steps are done: 1) we apply the usual tf.idf weighting scheme; 2) we delete words that appear too few times; 3) we normalized each document vector to have unit Euclidean length.

We tested three clustering algorithms: 1) p-QR, this refers to the algorithm using the eigenvector matrix followed by pivoted QR decomposition for cluster membership assignment; 2) p-Kmeans, we compute the eigenvector matrix, and then apply K-means on the rows of the eigenvector matrix; 3) K-means, this is K-means directly applied to the original data vectors. For both K-means methods, we start with a set of cluster centers chosen randomly from the (projected) data vectors, and we aslo make sure that the *same* random set is used for both for comparison. To assess the quality of a clustering algorithm, we take advantage of the fact that the newsgroup data are already labeled and we measure the performance by the accuracy of the clustering algorithm against the document category labels [10]. In particular, for a $k$ cluster case, we compute a $k$-by-$k$ confusion matrix $C = [c_{ij}]$ with $c_{ij}$ the number of documents in cluster $i$ that belongs to newsgroup category $j$. It is actually quite subtle to compute the accuracy using the confusion matrix because we do not know which cluster matches which newsgroup category. An optimal way is to solve the following maximization problem

$$\max\{ \text{ trace}(CP) \mid P \text{ is a permutation matrix}\},$$

and divide the maximum by the total number of documents to get the accuracy. This is equivalent to finding perfect matching a complete weighted bipartite graph, one can use Kuhn-Munkres algorithm [7]. In all our experiments, we used a greedy algorithm to compute a sub-optimal solution.

Table 1: Comparison of p-QR, p-Kmeans, and K-means for two-way clustering

| Newsgroups | p-QR | p-Kmeans | K-means |
|---|---|---|---|
| NG1/NG2 | $89.29 \pm 7.51\%$ | $89.62 \pm 6.90\%$ | $76.25 \pm 13.06\%$ |
| NG2/NG3 | $62.37 \pm 8.39\%$ | $63.84 \pm 8.74\%$ | $61.62 \pm 8.03\%$ |
| NG8/NG9 | $75.88 \pm 8.88\%$ | $77.64 \pm 9.00\%$ | $65.65 \pm 9.26\%$ |
| NG10/NG11 | $73.32 \pm 9.08\%$ | $74.86 \pm 8.89\%$ | $62.04 \pm 8.61\%$ |
| NG1/NG15 | $73.32 \pm 9.08\%$ | $74.86 \pm 8.89\%$ | $62.04 \pm 8.61\%$ |
| NG18/NG19 | $63.86 \pm 6.09\%$ | $64.04 \pm 7.23\%$ | $63.66 \pm 8.48\%$ |

Table 2: Comparison of p-QR, p-Kmeans, and K-means for multi-way clustering

| Newsgroups | | p-QR | p-Kmeans | K-means |
|---|---|---|---|---|
| NG2/NG3/NG4/NG5/NG6 (50) | | $40.36 \pm 5.17\%$ | $41.15 \pm 5.73\%$ | $35.77 \pm 5.19\%$ |
| NG2/NG3/NG4/NG5/NG6 (100) | | $41.67 \pm 5.06\%$ | $42.53 \pm 5.02\%$ | $37.20 \pm 4.39\%$ |
| NG2/NG9/NG10/NG15/NG18 (50) | | $77.83 \pm 9.26\%$ | $70.13 \pm 11.67\%$ | $58.10 \pm 9.60\%$ |
| NG2/NG9/NG10/NG15/NG18 (100) | | $79.91 \pm 9.90\%$ | $75.56 \pm 10.63\%$ | $66.37 \pm 10.89\%$ |
| NG1/NG5/NG7/NG8/NG11/ NG12/NG13/NG14/NG15/NG17 | (50) | $60.21 \pm 4.88\%$ | $58.18 \pm 4.41\%$ | $40.18 \pm 4.64\%$ |
| NG1/NG5/NG7/NG8/NG11/ NG12/NG13/NG14/NG15/NG17 | (100) | $65.08 \pm 5.14\%$ | $58.99 \pm 5.22\%$ | $48.33 \pm 5.64\%$ |

EXAMPLE 1. In this example, we look at binary clustering. We choose 50 random document vectors each from two newsgroups. We tested 100 runs for each pair of newsgroups, and list the means and standard deviations in Table 1. The two clustering algorithms p-QR and p-Kmeans are comparable to each other, and both are better and sometimes substantially better than K-means.

EXAMPLE 2. In this example, we consider $k$-way clustering with $k = 5$ and $k = 10$. Three newsgroup sets are chosen with 50 and 100 random samples from each newsgroup as indicated in the parenthesis. Again 100 runs are used for each tests and the means and standard deviations are listed in Table 2. Moreover, in Figure 1, we also plot the accuracy for the 100 runs for the test NG2/NG9/NG10/NG15/NG18 (50). Both p-QR and p-Kmeans perform better than Kmeans. For newsgroup sets with small overlaps, p-QR performs better than p-Kmeans. This might be explained by the fact that p-QR explores the special structure of the eigenvector matrix and is therefore more efficient. As a less thorough comparison with the *information bottleneck method* used in [10], there for 15 runs of NG2/NG9/NG10/NG15/NG18 (100) mean accuracy 56.67% with maximum accuracy 67.00% is obtained. For 15 runs of the 10 newsgroup set with 50 samples, mean accuracy 35.00% with maximum accuracy about 40.00% is obtained.

EXAMPLE 3. We compare the lower bound given in (4). We only list a typical sample from NG2/NG9/NG10/NG15/NG18 (50). The column with "NG labels" indicates clustering using the newsgroup labels and by definition has 100% accuracy. It is quite clear that the newsgroup categories are not completely captured by the sum-of-squares cost function because p-QR and "NG labels" both have higher accuracy but also larger sum-of-squares values. Interestingly, it seems that p-QR captures some of this information of the newsgroup categories.

| | p-QR | p-Kmeans | K-means | NG labels | lower bound |
|---|---|---|---|---|---|
| accuracy | 86.80% | 83.60% | 57.60% | 100% | N/A |
| ss($\Pi$) | 224.1110 | 223.8966 | 228.8416 | 224.4040 | 219.0266 |

## Acknowledgments

This work was supported in part by NSF grant CCR-9901986 and by Department of Energy through an LBL LDRD fund.

## Footnotes

[1]The newsgroup dataset together with the `bow` toolkit for processing it can be downloaded from `http://www.cs.cmu.edu/afs/cs/project/theo-11/www/naive-bayes.html`.

## References

[1] P. S. Bradley and Usama M. Fayyad. (1998). *Refining Initial Points for K-Means Clustering.* Proc. 15th International Conf. on Machine Learning, 91–99.

[2] P. S. Bradley, K. Bennett and A. Demiritz. *Constrained K-means Clustering.* Microsoft Research, MSR-TR-2000-65, 2000.

[3] M. Girolani. (2001). Mercer Kernel Based Clustering in Feature Space. To appear in *IEEE Transactions on Neural Networks.*

[4] G. Golub and C. Van Loan. (1996). *Matrix Computations.* Johns Hopkins University Press, 3rd Edition.

[5] Ming Gu, Hongyuan Zha, Chris Ding, Xiaofeng He and Horst Simon. (2001). *Spectral Embedding for K-Way Graph Clustering.* Technical Report, Department of Computer Science and Engineering, CSE-01-007, Pennsylvania State University.

[6] J.A. Hartigan and M.A. Wong. (1979). *A K-means Clustering Algorithm.* Applied Statistics, 28:100–108.

[7] L. Lovasz and M.D. Plummer. (1986) *Matching Theory.* Amsterdam: North Holland.

[8] A. McCallum. Bow: A toolkit for statistical language modeling, text retrieval, classification and clustering. `http://www.cs.cmu.edu/ mccallum/bow`.

[9] B. Schölkopf, A. Smola and K.R. Müller. (1998). Nonlinear Component Analysis as a Kernel Eigenvalue Problem. *Neural Computation*, 10: 1299–1219.

[10] N. Slonim and N. Tishby. (2000). *Document clustering using word clusters via the information bottleneck method.* Proceedings of SIGIR-2000.

[11] G.W. Stewart and J.G. Sun. (1990). *Matrix Perturbation Theory.* Academic Press, San Diego, CA.
